# Illumination and View Position in 3D Visual Recognition

**Amnon Shashua**
M.I.T. Artificial Intelligence Lab., NE43-737
and Department of Brain and Cognitive Science
Cambridge, MA 02139

## Abstract

It is shown that both changes in viewing position and illumination conditions can be compensated for, prior to recognition, using combinations of images taken from different viewing positions and different illumination conditions. It is also shown that, in agreement with psychophysical findings, the computation requires at least a sign–bit image as input — contours alone are not sufficient.

## 1 Introduction

The task of visual recognition is natural and effortless for biological systems, yet the problem of recognition has been proven to be very difficult to analyze from a computational point of view. The fundamental reason is that novel images of familiar objects are often not sufficiently similar to previously seen images of that object. Assuming a rigid and isolated object in the scene, there are two major sources for this variability: geometric and photometric. The geometric source of variability comes from changes of view position. A 3D object can be viewed from a variety of directions, each resulting with a different 2D projection. The difference is significant, even for modest changes in viewing positions, and can be demonstrated by superimposing those projections (see Fig. 4, first row second image). Much attention has been given to this problem in the visual recognition literature ([9], and references therein), and recent results show that one can compensate for changes in viewing position by generating novel views from a small number of model views of the object [10, 4, 8].

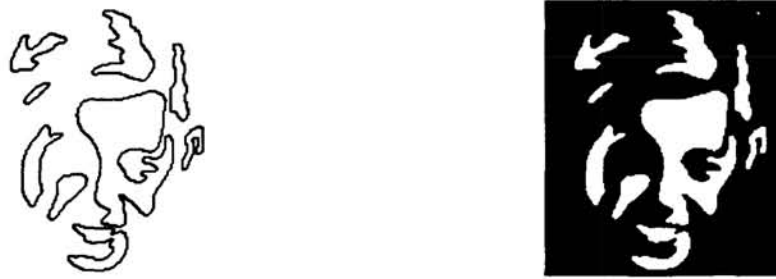

Figure 1: A 'Mooney' image. See text for details.

The photometric source of variability comes from changing illumination conditions (positions and distribution of light sources in the scene). This has the effect of changing the brightness distribution in the image, and the location of shadows and specular reflections. The traditional approach to this problem is based on the notion of edge detection. The idea is that discontinuities in image brightness remain stable under changes of illumination conditions. This invariance is not complete and furthermore it is an open question whether this kind of contour information is sufficient, or even relevant, for purposes of visual recognition.

Consider the image in Fig. 1, adopted from Mooney's Closure Faces Test [6]. Most observers show no difficulty in interpreting the shape of the object from the right-hand image, but cannot identify the object when presented with only the contours. Also, many of the contours are shadow contours and therefore critically rely on the direction of light source. In Fig. 2 four frontal images of a doll from four different illumination conditions are shown together with their intensity step edges. The change in the contour image is significant and is not limited to shadow contours — some object edges appear or disappear as a result of the change in brightness distribution. Also shown in Fig. 4 is a sign–bit image of the intensity image followed by a convolution with a Difference of Gaussians. As with the Mooney image, it is considerably more difficult to interpret the image of a complex object with only the zero–crossing (or level–crossing) contours than when the sign–bits are added.

It seems, therefore, that a successful recognition scheme should be able to cope with changes in illumination conditions, as well as changes in viewing positions, by working with a richer source of information than just contours (for a different point of view, see [1]). The minimal information that seems to be sufficient, at least for coping with the photometric problem, is the sign–bit image.

The approach to visual recognition in this study is in line with the 'alignment' approach [9] and is also inspired by the work of Ullman and Basri [10] who show that the geometric source of variability can be handled by matching the novel projection to a linear combination of a small number of previously seen projections of that object. A recognition scheme that can handle both the geometric and photometric sources of variability is suggested by introducing three new results: (i) any image of a surface with a linear reflectance function (including Lambertian and Phong's model without point specularities) can be expressed as a linear combination of a fixed set of three images of that surface taken under different illumination conditions, (ii) from a computational standpoint, the coefficients are better recovered using the

sign–bit image rather than the contour image, and (iii) one can compensate for both changes in viewing position and illumination conditions by using combinations of images taken from different viewing positions and different illumination conditions.

## 2    Linear Combination of Images

We start by assuming that view position is fixed and the only parameter that is allowed to change is the positions and distribution of light sources. The more general result that includes changes in viewing positions will be discussed in section 4.

**Proposition 1** *All possible images of a surface, with a linear reflectance function, generated by all possible illumination conditions (positions and distribution of light sources) are spanned by a linear combination of images of the surface taken from independent illumination conditions.*

**Proof:**   Follows directly from the general result that if $f_j(x)$, $x \in R^k$, $j = 1, ..., k$, are $k$ linear functions, which are also linearly independent, then for any linear function $f(x)$, we have that $f(x) = \sum_j a_j f_j(x)$, for some constants $a_j$. $\Box$

The simplest case for which this result holds is the Lambertian reflectance model under a point light source (observed independently by Yael Moses, personal communication). Let $r$ be an object point projecting to $p$. Let $n_r$ represent the normal and albedo at $r$ (direction and magnitude), and $s$ represent the light source and its intensity. The brightness at $p$ under the Lambertian model is $I(p) = n_r \cdot s$, and because $s$ is fixed for all point $p$, we have $I(p) = a_1 I_1(p) + a_2 I_2(p) + a_3 I_3(p)$ where $I_j(p)$ is the brightness under light source $s_j$ and where $s_1, s_2, s_3$ are linearly independent. This result generalizes, in a straightforward manner, to the case of multiple light sources as well.

The Lambertian model is suitable for matte surfaces, i.e. surfaces that diffusely reflect incoming light rays. One can add a 'shininess' component to account for the fact that for non-ideal Lambertian surfaces, more light is reflected in a direction making an equal angle of incidence with reflectance. In Phong's model of reflectance [7] this takes the form of $(n_r \cdot h)^c$ where $h$ is the bisector of $s$ and the viewer's direction $v$. The power constant $c$ controls the degree of sharpness of the point specularity, therefore outside that region one can use a linear version of Phong's model by replacing the power constant with a multiplicative constant, to get the following function: $I(p) = n_r \cdot [s + \rho(v + s)]$. As before, the bracketed vector is fixed for all image points and therefore the linear combination result holds.

The linear combination result suggests therefore that changes in illumination can be compensated for, prior to recognition, by selecting three points (that are visible to $s, s_1, s_2, s_3$) to solve for $a_1, a_2, a_3$ and then match the novel image $I$ with $I' = \sum_j a_j I_j$. The two images should match along all points $p$ whose object points $r$ are visible to $s_1, s_2, s_3$ (even if $n_r \cdot s < 0$, i.e. $p$ is attached–shadowed); approximately match along points for which $n_r \cdot s_j < 0$, for some $j$ ($I_j(p)$ is truncated to zero, geometrically $s$ is projected onto the subspace spanned by the remaining basis light sources) and not match along points that are cast–shadowed in $I$ ($n_r \cdot s > 0$ but $r$ is not visible to $s$ because of self occlusion). Coping with cast–shadows is an important task, but is not in the scope of this paper.

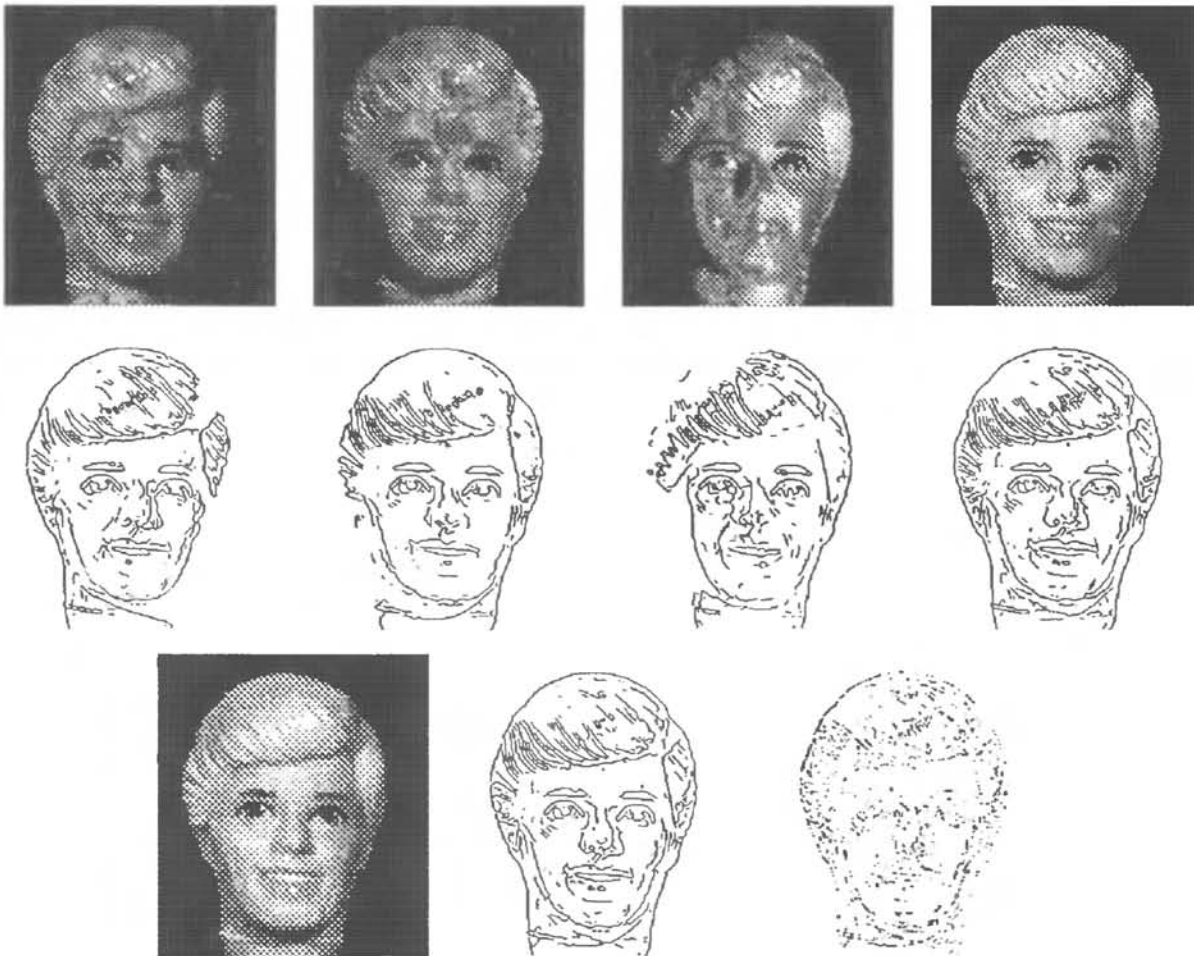

Figure 2: Linear combination of model images taken from the same viewing position and under different illumination conditions. *Row 1,2:* Three model images taken under a varying point light source, and the input image, and their brightness edges. *Row 3:* The image generated by the linear combination of the model images, its edges, and the difference edge image between the input and generated image.

The linear combination result also implies that, for the purposes of recognition, one does not need to recover shape or light source direction in order to compensate for changes in brightness distribution and attached shadows. Experimental results, on a non-ideal Lambertian surface, are shown in Fig. 2.

## 3   Coefficients from Contours and Sign–bits

Mooney pictures, such as in Fig. 1, demonstrate that humans can cope well with situations of varying illumination by using only limited information from the input image, namely the sign–bits, yet are not able to do so from contours alone. This observation can be predicted from a computational standpoint, as shown below.

**Proposition 2** *The coefficients that span an image I by the basis of three other images, as described in proposition 1, can be solved, up to a common scale factor,*

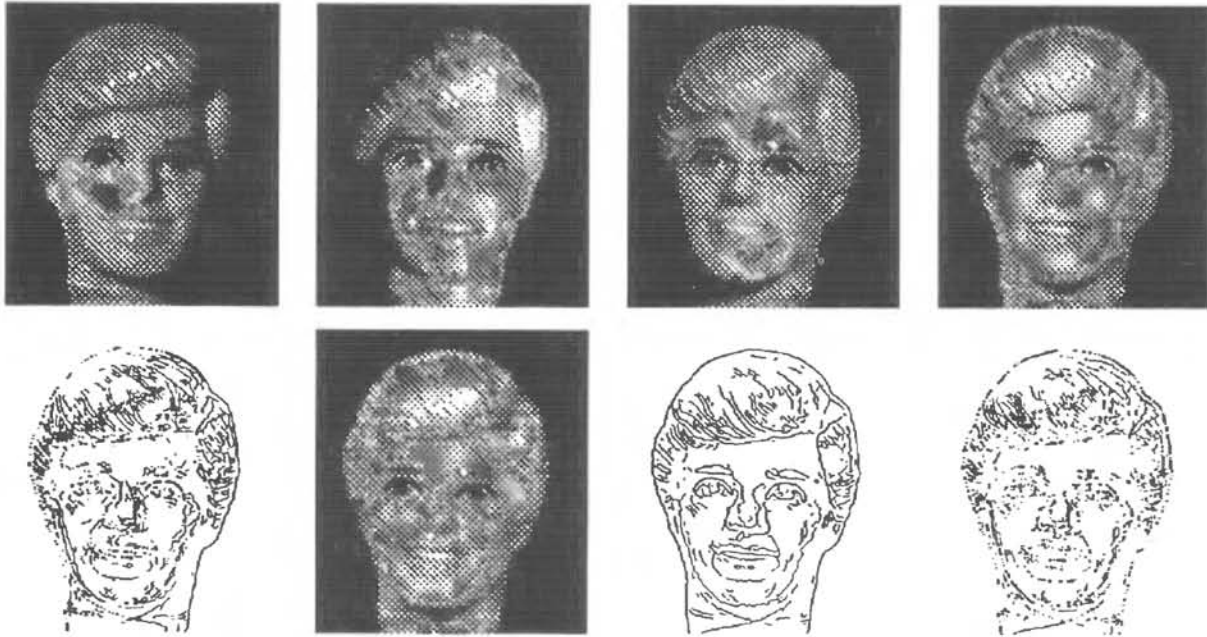

Figure 3: Compensating for both changes in view and illumination. *Row 1:* Three model images, one of which is taken from a different viewing direction (23° apart), and the input image from a novel viewing direction (in between the model images) and illumination condition. *Row 2:* difference image between the edges of the input image (shown separately in Fig. 4) and the edges of the view transformed first model image (first row, lefthand), the final generated image (linear combination of the three transformed model images), its edges, and the difference image between edges of input and generated image.

*from just the contours of $I$ — zero-crossings or level-crossings.*

**Proof:** Let $a_j$ be the coefficients that span $I$ by the basis images $I_j$, $j = 1, 2, 3$, i.e. $I = \sum_j a_j I_j$. Let $f, f_j$ be the result of applying a Difference of Gaussians (DOG) operator, with the same scale, on images $I, I_j$, $j = 1, 2, 3$. Since DOG is a linear operator we have that $f = \sum_j a_j f_j$. Since $f(p) = 0$ along zero-crossing points $p$ of $I$, then by taking any three zero-crossing points, which are not on a cast–shadow border, we get a homogeneous set of equations from which $a_j$ can be solved up to a common scale factor.

Similarly, let $k$ be an unknown threshold applied to $I$. Therefore, along level crossings of $I$ we have $k = \sum_j a_j I_j$, hence 4 level-crossing points, that are visible to all four light sources, are sufficient to solve for $a_j$ and $k$. ☐

This result is in accordance with what is known from image compression literature of reconstructing an image, up to a scale factor, from contours alone [2]. In both cases, here and in image compression, this result may be difficult to apply in practice because the contours are required to be given at sub-pixel accuracy. One can relax the accuracy requirement by using the gradients along the contours — a technique that works well in practice. Nevertheless, neither gradients nor contours at sub-pixel accuracy are provided by Mooney pictures, which leaves us with the sign–bits as the source of information for solving for the coefficients.

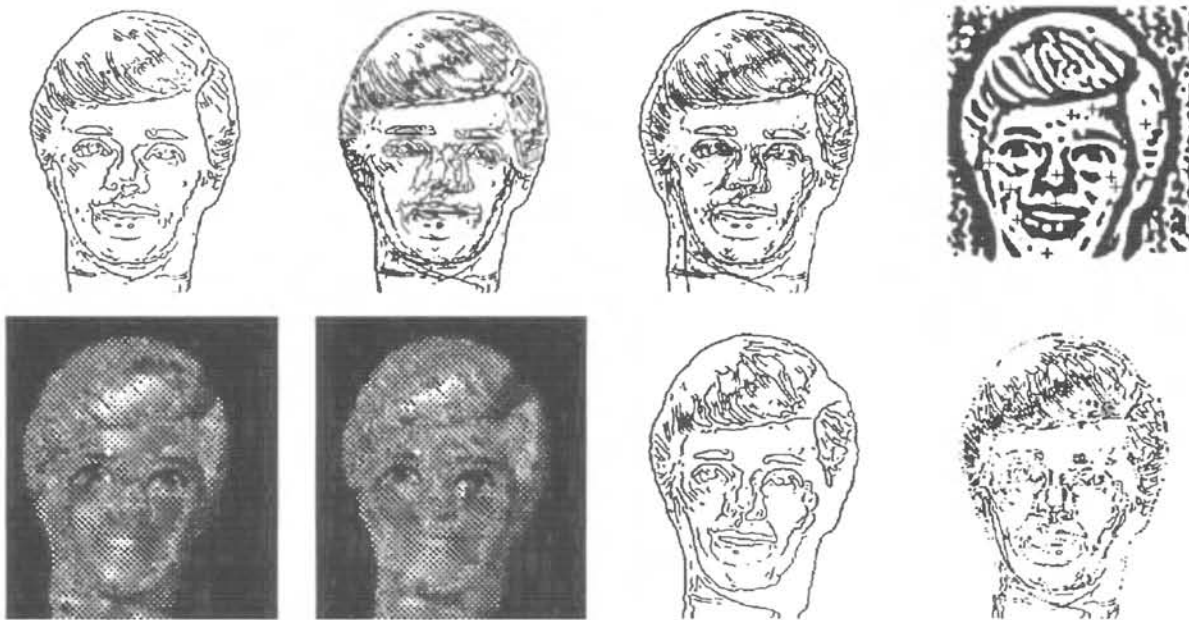

Figure 4: Compensating for changes in viewing position and illumination from a single view (model images are all from a single viewing position). Model images are the same as in Fig. 2, input image the same as in Fig. 3. *Row 1:* edges of input image, overlay of input edge image and edges of first model image, overlay with edges of the 2D affine transformed first model image, sign–bit input image with marked 'example' locations (16 of them). *Row 2:* linear combination image of the 2D affine transformed model images, the final generated image, its edges, overlay with edges of the input image.

**Proposition 3** *Solving for the coefficients from the sign–bit image of I is equivalent to solving for a separating hyperplane in 3D in which image points serve as 'examples'.*

**Proof:** Let $z(p) = (f_1, f_2, f_3)^T$ be a vector function and $\omega = (a_1, a_2, a_3)^T$ be the unknown weight vector. Given the sign-bit image $\hat{f}$ of $I$, we have that for every point $p$, excluding zero-crossings, the scalar product $\omega^T z(p)$ is either positive or negative. In this respect, one can consider points in $\hat{f}$ as 'examples' in 3D space and the coefficients $a_j$ as a vector normal to the separating hyperplane. □

A similar result can be obtained for the case of a thresholded image. The separating hyperplane in that case is defined in 4D, rather than 3D. Many schemes for finding a separating hyperplane have been described in Neural Network literature (see [5] for review) and in Discriminant Analysis literature ([3], for example). Experimental results shown in the next section show that 10—20 points, distributed over the entire object, are sufficient to produce results that are indistinguishable from those obtained from an exact solution.

By using the sign-bits instead of the zero-crossing contours we are trading a unique (up to a scale factor), but unstable, solution for an approximate, but stable, one. Also, by taking the sample points relatively far away from the contours (in order to minimize the chance of error) the scheme can tolerate a certain degree of misalign-

ment between the basis images and the novel image. This property will be used in one of the schemes, described below, for combining changes of viewing positions and illumination conditions.

## 4    Changing Illumination and Viewing Positions

In this section, the recognition scheme is generalized to cope with both changes in illumination and viewing positions. Namely, given a set of images of an object as a model and an input image viewed from a novel viewing position and taken under a novel illumination condition we would like to generate an image, from the model, that is similar to the input image.

**Proposition 4** *Any set of three images, satisfying conditions of proposition 1, of an object can be used to compensate for both changes in view and illumination.*

**Proof:** Any change in viewing position will induce both a change in the location of points in the image, and a change in their brightness (because of change in viewing angle and change in angle between light source and surface normal). From proposition 1, the change in brightness can be compensated for provided all the images are in alignment. What remains, therefore, is to bring the model images and the input image into alignment.

*Case 1:* If each of the three model images is viewed from a different position, then the remaining proof follows directly from the result of Ullman and Basri [10] who show that any view of an object with smooth boundaries, undergoing any affine transformation in space, is spanned by three views of the object.

*Case 2:* If only two of the model images are viewed from different positions, then given full correspondence between all points in the two model views and 4 corresponding points with the input image, we can transform all three model images to align with the input image in the following way. The 4 corresponding points between the input image and one of the model images define three corresponding vectors (taking one of the corresponding points, say $o$, as an origin) from which a 2D affine transformation, matrix $A$ and vector $w$, can be recovered. The result, proved in [8], is that for every point $p'$ in the input image who is in correspondence with $p$ in the model image we have that $p' = [Ap + o' - Ao] + \alpha_p w$. The parameter $\alpha_p$ is invariant to any affine transformation in space, therefore is also invariant to changes in viewing position. One can, therefore, recover $\alpha_p$ from the known correspondence between two model images and use that to predict the location $p'$. It can be shown that this scheme provides also a good approximation in the case of objects with smooth boundaries (like an egg or a human head, for details see [8]).

*Case 3:* All three model images are from the same viewing position. The model images are first brought into 'rough alignment' (term adopted from [10]) with the input image by applying the transformation $Ap + o' - Ao + w$ to all points $p$ in each model image. The remaining displacement between the transformed model images and the input image is $(\alpha_p - 1)w$ which can be shown to be bounded by the depth variation of the surface [8]. (In case the object is not sufficiently flat, more than 4 points may be used to define local transformations via a triangulation of those points). The linear combination coefficients are then recovered using the sign–bit

scheme described in the previous section. The three transformed images are then linearly combined to create a new image that is compensated for illumination but is still displaced from the input image. The displacement can be recovered by using a brightness correlation scheme along the direction $w$ to find $\alpha_p - 1$ for each point $p$. (for details, see [8]). []

Experimental results of the last two schemes are shown in Figs. 3 and 4. The four corresponding points, required for view compensation, were chosen manually along the tip of eyes, eye-brow and mouth of the doll. The full correspondence that is required between the third model view and the other two in scheme 2 above, was established by first taking two pictures of the third view, one from a novel illumination condition and the other from a similar illumination condition to one of the other model images. Correspondence was then determined by using the scheme described in [8]. The extra picture was then discarded. The sample points for the linear combination were chosen automatically by selecting 10 points in smooth brightness regions. The sample points using the sign–bit scheme were chosen manually.

## 5   Summary

It has been shown that the effects photometry and geometry in visual recognition can be decoupled and compensated for prior to recognition. Three new results were shown: (i) photometric effects can be compensated for using a linear combination of images, (ii) from a computational standpoint, contours alone are not sufficient for recognition, and (iii) geometrical effects can be compensated for from any set of three images, from different illuminations, of the object.

### Acknowledgments

I thank Shimon Ullman for his advice and support. Thanks to Ronen Basri, Tomaso Poggio, Whitman Richards and Daphna Weinshall for many discussions. A.S. is supported by NSF grant IRI-8900267.

### References

[1] Cavanagh,P. *Proc. 13th ECVP, Andrei,G. (Ed.), 1990.*

[2] Curtis,S.R and Oppenheim,A.V. in Whitman,R. and Ullman,S. (eds.) *Image Understanding 1989.* pp.92–110, Ablex, NJ 1990.

[3] Duda,R.O. and Hart,P.E. *pattern classification and scene analysis.* NY, Wiley 1973.

[4] Edelman,S. and Poggio,T. *Massachusetts Institute of Technology, A.I. Memo 1181, 1990*

[5] Lippmann,R.P. *IEEE ASSP Magazine,* pp.4–22, 1987.

[6] Mooney,C.M. *Can. J. Psychol.* 11:219–226, 1957.

[7] Phong,B.T. *Comm. ACM,* 18, 6:311-317, 1975.

[8] Shashua,A. *Massachusetts Institute of Technology, A.I. Memo 1327, 1991*

[9] Ullman,S. *Cognition,*32:193–254, 1989.

[10] Ullman,S. and Basri,R. *Massachusetts Institute of Technology, A.I. Memo 1052, 1989*